# One Permutation Hashing

**Ping Li**
Department of Statistical Science
Cornell University

**Art B Owen**
Department of Statistics
Stanford University

**Cun-Hui Zhang**
Department of Statistics
Rutgers University

## Abstract

Minwise hashing is a standard procedure in the context of search, for efficiently estimating set similarities in massive binary data such as text. Recently, $b$-bit minwise hashing has been applied to large-scale learning and sublinear time near-neighbor search. The major drawback of minwise hashing is the expensive pre-processing, as the method requires applying (e.g.,) $k = 200$ to $500$ permutations on the data. This paper presents a simple solution called *one permutation hashing*. Conceptually, given a binary data matrix, we permute the columns once and divide the permuted columns evenly into $k$ bins; and we store, for each data vector, the smallest nonzero location in each bin. The probability analysis illustrates that this one permutation scheme should perform similarly to the original ($k$-permutation) minwise hashing. Our experiments with training SVM and logistic regression confirm that one permutation hashing can achieve similar (or even better) accuracies compared to the $k$-permutation scheme. *See more details in arXiv:1208.1259.*

## 1 Introduction

Minwise hashing [4, 3] is a standard technique in the context of search, for efficiently computing set similarities. Recently, $b$-bit minwise hashing [18, 19], which stores only the lowest $b$ bits of each hashed value, has been applied to sublinear time near neighbor search [22] and learning [16], on large-scale high-dimensional binary data (e.g., text). A drawback of minwise hashing is that it requires a costly preprocessing step, for conducting (e.g.,) $k = 200 \sim 500$ permutations on the data.

### 1.1 Massive High-Dimensional Binary Data

In the context of search, text data are often processed to be binary in extremely high dimensions. A standard procedure is to represent documents (e.g., Web pages) using $w$-shingles (i.e., $w$ contiguous words), where $w \geq 5$ in several studies [4, 8]. This means the size of the dictionary needs to be substantially increased, from (e.g.,) $10^5$ common English words to $10^{5w}$ "super-words". In current practice, it appears sufficient to set the total dimensionality to be $D = 2^{64}$, for convenience. Text data generated by $w$-shingles are often treated as binary. The concept of shingling can be naturally extended to Computer Vision, either at pixel level (for aligned images) or at Visual feature level [23].

In machine learning practice, the use of extremely high-dimensional data has become common. For example, [24] discusses training datasets with (on average) $n = 10^{11}$ items and $D = 10^9$ distinct features. [25] experimented with a dataset of potentially $D = 16$ trillion ($1.6 \times 10^{13}$) unique features.

### 1.2 Minwise Hashing and $b$-Bit Minwise Hashing

Minwise hashing was mainly designed for binary data. A binary (0/1) data vector can be viewed as a set (locations of the nonzeros). Consider sets $S_i \subseteq \Omega = \{0, 1, 2, ..., D-1\}$, where $D$, the size of the space, is often set as $D = 2^{64}$ in industrial applications. The similarity between two sets, $S_1$ and $S_2$, is commonly measured by the *resemblance*, which is a version of the normalized inner product:

$$R = \frac{|S_1 \cap S_2|}{|S_1 \cup S_2|} = \frac{a}{f_1 + f_2 - a}, \quad \text{where } f_1 = |S_1|, \ f_2 = |S_2|, \ a = |S_1 \cap S_2| \tag{1}$$

For large-scale applications, the cost of computing resemblances exactly can be prohibitive in time, space, and energy-consumption. The minwise hashing method was proposed for efficient computing resemblances. The method requires applying $k$ independent random permutations on the data.

Denote $\pi$ a random permutation: $\pi : \Omega \to \Omega$. The hashed values are the two minimums of $\pi(S_1)$ and $\pi(S_2)$. The probability at which the two hashed values are equal is

$$\mathbf{Pr}\left(\min(\pi(S_1)) = \min(\pi(S_2))\right) = \frac{|S_1 \cap S_2|}{|S_1 \cup S_2|} = R \tag{2}$$

One can then estimate $R$ from $k$ independent permutations, $\pi_1, ..., \pi_k$:

$$\hat{R}_M = \frac{1}{k}\sum_{j=1}^{k} 1\{\min(\pi_j(S_1)) = \min(\pi_j(S_2))\}, \qquad \text{Var}\left(\hat{R}_M\right) = \frac{1}{k}R(1-R) \qquad (3)$$

Because the indicator function $1\{\min(\pi_j(S_1)) = \min(\pi_j(S_2))\}$ can be written as an inner product between two binary vectors (each having only one 1) in $D$ dimensions [16]:

$$1\{\min(\pi_j(S_1)) = \min(\pi_j(S_2))\} = \sum_{i=0}^{D-1} 1\{\min(\pi_j(S_1)) = i\} \times 1\{\min(\pi_j(S_2)) = i\} \qquad (4)$$

we know that minwise hashing can be potentially used for training linear SVM and logistic regression on high-dimensional binary data by converting the permuted data into a new data matrix in $D \times k$ dimensions. This of course would not be realistic if $D = 2^{64}$.

The method of $b$-bit minwise hashing [18, 19] provides a simple solution by storing only the lowest $b$ bits of each hashed data, reducing the dimensionality of the (expanded) hashed data matrix to just $2^b \times k$. [16] applied this idea to large-scale learning on the *webspam* dataset and demonstrated that using $b = 8$ and $k = 200$ to 500 could achieve very similar accuracies as using the original data.

## 1.3   The Cost of Preprocessing and Testing

Clearly, the preprocessing of minwise hashing can be very costly. In our experiments, loading the *webspam* dataset (350,000 samples, about 16 million features, and about 24GB in Libsvm/svmlight (text) format) used in [16] took about 1000 seconds when the data were stored in text format, and took about 150 seconds after we converted the data into binary. In contrast, the preprocessing cost for $k = 500$ was about 6000 seconds. Note that, compared to industrial applications [24], the *webspam* dataset is very small. For larger datasets, the preprocessing step will be much more expensive.

In the testing phrase (in search or learning), if a new data point (e.g., a new document or a new image) has not been processed, then the total cost will be expensive if it includes the preprocessing. This may raise significant issues in user-facing applications where the testing efficiency is crucial.

Intuitively, the standard practice of minwise hashing ought to be very "wasteful" in that all the nonzero elements in one set are scanned (permuted) but only the smallest one will be used.

## 1.4   Our Proposal: One Permutation Hashing

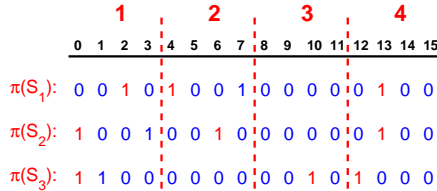

Figure 1: Consider $S_1, S_2, S_3 \subseteq \Omega = \{0, 1, ..., 15\}$ (i.e., $D = 16$). We apply one permutation $\pi$ on the sets and present $\pi(S_1)$, $\pi(S_2)$, and $\pi(S_3)$ as binary (0/1) vectors, where $\pi(S_1) = \{2, 4, 7, 13\}$, $\pi(S_2) = \{0, 6, 13\}$, and $\pi(S_3) = \{0, 1, 10, 12\}$. We divide the space $\Omega$ evenly into $k = 4$ bins, select the smallest nonzero in each bin, and **re-index** the selected elements as: $[2, 0, *, 1]$, $[0, 2, *, 1]$, and $[0, *, 2, 0]$. For now, we use '*' for empty bins, which occur rarely unless the number of nonzeros is small compared to $k$.

As illustrated in Figure 1, the idea of *one permutation hashing* is simple. We view sets as 0/1 vectors in $D$ dimensions so that we can treat a collection of sets as a binary data matrix in $D$ dimensions. After we permute the columns (features) of the data matrix, we divide the columns evenly into $k$ parts (bins) and we simply take, for each data vector, the smallest nonzero element in each bin.

In the example in Figure 1 (which concerns 3 sets), the sample selected from $\pi(S_1)$ is $[2, 4, *, 13]$, where we use '*' to denote an empty bin, for the time being. Since only want to compare elements with the same bin number (so that we can obtain an inner product), we can actually re-index the elements of each bin to use the smallest possible representations. For example, for $\pi(S_1)$, after re-indexing, the sample $[2, 4, *, 13]$ becomes $[2 - 4 \times 0, 4 - 4 \times 1, *, 13 - 4 \times 3] = [2, 0, *, 1]$.

We will show that empty bins occur rarely unless the total number of nonzeros for some set is small compared to $k$, and we will present strategies on how to deal with empty bins should they occur.

## 1.5 Advantages of One Permutation Hashing

Reducing $k$ (e.g., 500) permutations to just one permutation (or a few) is much more computationally efficient. From the perspective of energy consumption, this scheme is desirable, especially considering that minwise hashing is deployed in the search industry. Parallel solutions (e.g., GPU [17]), which require additional hardware and software implementation, will not be energy-efficient.

In the testing phase, if a new data point (e.g., a new document or a new image) has to be first processed with $k$ permutations, then the testing performance may not meet the demand in, for example, user-facing applications such as search or interactive visual analytics.

One permutation hashing should be easier to implement, from the perspective of random number generation. For example, if a dataset has one billion features ($D = 10^9$), we can simply generate a "permutation vector" of length $D = 10^9$, the memory cost of which (i.e., 4GB) is not significant. On the other hand, it would not be realistic to store a "permutation matrix" of size $D \times k$ if $D = 10^9$ and $k = 500$; instead, one usually has to resort to approximations such as universal hashing [5]. Universal hashing often works well in practice although theoretically there are always worst cases.

One permutation hashing is a better matrix sparsification scheme. In terms of the original binary data matrix, the one permutation scheme simply makes many nonzero entries be zero, without further "damaging" the matrix. Using the $k$-permutation scheme, we store, for each permutation and each row, only the first nonzero and make all the other nonzero entries be zero; and then we have to concatenate $k$ such data matrices. This significantly changes the structure of the original data matrix.

## 1.6 Related Work

One of the authors worked on another "one permutation" scheme named *Conditional Random Sampling (CRS)* [13, 14] since 2005. Basically, CRS continuously takes the bottom-$k$ nonzeros after applying one permutation on the data, then it uses a simple "trick" to construct a random sample for each pair with the effective sample size determined at the estimation stage. By taking the nonzeros continuously, however, the samples are no longer "aligned" and hence we can not write the estimator as an inner product in a unified fashion. [16] commented that using CRS for linear learning does not produce as good results compared to using $b$-bit minwise hashing. Interestingly, in the original "minwise hashing" paper [4] (we use quotes because the scheme was not called "minwise hashing" at that time), only one permutation was used and a sample was the first $k$ nonzeros after the permutation. Then they quickly moved to the $k$-permutation minwise hashing scheme [3].

We are also inspired by the work on *very sparse random projections* [15] and *very sparse stable random projections* [12]. The regular random projection method also has the expensive preprocessing cost as it needs a large number of projections. [15, 12] showed that one can substantially reduce the preprocessing cost by using an extremely sparse projection matrix. The preprocessing cost of very sparse random projections can be as small as merely doing one projection. See `www.stanford.edu/group/mmds/slides2012/s-pli.pdf` for the experimental results on clustering/classification/regression using very sparse random projections.

This paper focuses on the "fixed-length" scheme as shown in Figure 1. The technical report (arXiv:1208.1259) also describes a "variable-length" scheme. Two schemes are more or less equivalent, although the fixed-length scheme is more convenient to implement (and it is slightly more accurate). The variable-length hashing scheme is to some extent related to the Count-Min (CM) sketch [6] and the Vowpal Wabbit (VW) [21, 25] hashing algorithms.

## 2 Applications of Minwise Hashing on Efficient Search and Learning

In this section, we will briefly review two important applications of the $k$-permutation $b$-bit minwise hashing: (i) sublinear time near neighbor search [22], and (ii) large-scale linear learning [16].

### 2.1 Sublinear Time Near Neighbor Search

The task of *near neighbor search* is to identify a set of data points which are "most similar" to a query data point. Developing efficient algorithms for near neighbor search has been an active research topic since the early days of modern computing (e.g, [9]). In current practice, methods for approximate near neighbor search often fall into the general framework of *Locality Sensitive Hashing (LSH)* [10, 1]. The performance of LSH largely depends on its underlying implementation. The idea in [22] is to directly use the bits from $b$-bit minwise hashing to construct hash tables.

Specifically, we hash the data points using $k$ random permutations and store each hash value using $b$ bits. For each data point, we concatenate the resultant $B = bk$ bits as a *signature* (e.g., $bk = 16$). This way, we create a table of $2^B$ buckets and each bucket stores the pointers of the data points whose signatures match the bucket number. In the testing phrase, we apply the same $k$ permutations to a query data point to generate a $bk$-bit signature and only search data points in the corresponding bucket. Since using only one table will likely miss many true near neighbors, as a remedy, we independently generate $L$ tables. The query result is the union of data points retrieved in $L$ tables.

| Index | | Data Points | | Index | | Data Points |
|---|---|---|---|---|---|---|
| 00 | 00 | *6*, 110, 143 | | 00 | 00 | 8, 159, 331 |
| 00 | 01 | 3, 38, 217 | | 00 | 01 | 11, 25, 99 |
| 00 | 10 | (empty) | | 00 | 10 | 3, 14, 32, 97 |
| | | | | | | |
| 11 | 01 | 5, 14, 206 | | 11 | 01 | 7, 49, 208 |
| 11 | 10 | 31, 74, 153 | | 11 | 10 | 33, 489 |
| 11 | 11 | 21, 142, 329 | | 11 | 11 | *6*, 15, 26, 79 |

Figure 2: An example of hash tables, with $b = 2$, $k = 2$, and $L = 2$.

Figure 2 provides an example with $b = 2$ bits, $k = 2$ permutations, and $L = 2$ tables. The size of each hash table is $2^4$. Given $n$ data points, we apply $k = 2$ permutations and store $b = 2$ bits of each hashed value to generate $n$ (4-bit) signatures $L$ times. Consider data point 6. For Table 1 (left panel of Figure 2), the lowest $b$-bits of its two hashed values are 00 and 00 and thus its signature is 0000 in binary; hence we place a pointer to data point 6 in bucket number 0. For Table 2 (right panel of Figure 2), we apply another $k = 2$ permutations. This time, the signature of data point 6 becomes 1111 in binary and hence we place it in the last bucket. Suppose in the testing phrase, the two (4-bit) signatures of a new data point are 0000 and 1111, respectively. We then only search for the near neighbors in the set $\{6, 15, 26, 79, 110, 143\}$, instead of the original set of $n$ data points.

## 2.2 Large-Scale Linear Learning

The recent development of highly efficient linear learning algorithms is a major breakthrough. Popular packages include SVM$^{\text{perf}}$ [11], Pegasos [20], Bottou's SGD SVM [2], and LIBLINEAR [7].

Given a dataset $\{(\mathbf{x}_i, y_i)\}_{i=1}^n$, $\mathbf{x}_i \in \mathbb{R}^D$, $y_i \in \{-1, 1\}$, the $L_2$-regularized logistic regression solves the following optimization problem (where $C > 0$ is the regularization parameter):

$$\min_{\mathbf{w}} \quad \frac{1}{2}\mathbf{w}^{\mathbf{T}}\mathbf{w} + C\sum_{i=1}^n \log\left(1 + e^{-y_i\mathbf{w}^{\mathbf{T}}\mathbf{x_i}}\right), \tag{5}$$

The $L_2$-regularized linear SVM solves a similar problem:

$$\min_{\mathbf{w}} \quad \frac{1}{2}\mathbf{w}^{\mathbf{T}}\mathbf{w} + C\sum_{i=1}^n \max\left\{1 - y_i\mathbf{w}^{\mathbf{T}}\mathbf{x_i}, \, 0\right\}, \tag{6}$$

In [16], they apply $k$ random permutations on each (binary) feature vector $\mathbf{x}_i$ and store the lowest $b$ bits of each hashed value, to obtain a new dataset which can be stored using merely $nbk$ bits. At run-time, each new data point has to be expanded into a $2^b \times k$-length vector with exactly $k$ 1's.

To illustrate this simple procedure, [16] provided a toy example with $k = 3$ permutations. Suppose for one data vector, the hashed values are $\{12013, 25964, 20191\}$, whose binary digits are respectively $\{010111011101101, 110010101101100, 100111011011111\}$. Using $b = 2$ bits, the binary digits are stored as $\{01, 00, 11\}$ (which corresponds to $\{1, 0, 3\}$ in decimals). At run-time, the ($b$-bit) hashed data are expanded into a new feature vector of length $2^b k = 12$: $\{0, 0, 1, 0, \ 0, 0, 0, 1, \ 1, 0, 0, 0\}$. The same procedure is then applied to all $n$ feature vectors.

Clearly, in both applications (near neighbor search and linear learning), the hashed data have to be "aligned" in that only the hashed data generated from the same permutation are interacted. Note that, with our one permutation scheme as in Figure 1, the hashed data are indeed aligned.

## 3 Theoretical Analysis of the One Permutation Scheme

This section presents the probability analysis to provide a rigorous foundation for one permutation hashing as illustrated in Figure 1. Consider two sets $S_1$ and $S_2$. We first introduce two definitions,

for the number of "jointly empty bins" and the number of "matched bins," respectively:

$$N_{emp} = \sum_{j=1}^{k} I_{emp,j}, \qquad N_{mat} = \sum_{j=1}^{k} I_{mat,j} \qquad (7)$$

where $I_{emp,j}$ and $I_{mat,j}$ are defined for the $j$-th bin, as

$$I_{emp,j} = \begin{cases} 1 & \text{if both } \pi(S_1) \text{ and } \pi(S_2) \text{ are empty in the } j\text{-th bin} \\ 0 & \text{otherwise} \end{cases} \qquad (8)$$

$$I_{mat,j} = \begin{cases} 1 & \text{if both } \pi(S_1) \text{ and } \pi(S_1) \text{ are not empty and the smallest element of } \pi(S_1) \\ & \quad \text{matches the smallest element of } \pi(S_2), \text{ in the } j\text{-th bin} \\ 0 & \text{otherwise} \end{cases} \qquad (9)$$

Recall the notation: $f_1 = |S_1|$, $f_2 = |S_2|$, $a = |S_1 \cap S_2|$. We also use $f = |S_1 \cup S_2| = f_1 + f_2 - a$.

**Lemma 1**

$$\mathbf{Pr}(N_{emp} = j) = \sum_{s=0}^{k-j} (-1)^s \frac{k!}{j!s!(k-j-s)!} \prod_{t=0}^{f-1} \frac{D\left(1 - \frac{j+s}{k}\right) - t}{D - t}, \ \ 0 \le j \le k-1 \qquad (10)$$

*Assume* $D\left(1 - \frac{1}{k}\right) \ge f = f_1 + f_2 - a.$

$$\frac{E(N_{emp})}{k} = \prod_{j=0}^{f-1} \frac{D\left(1 - \frac{1}{k}\right) - j}{D - j} \le \left(1 - \frac{1}{k}\right)^f \qquad (11)$$

$$\frac{E(N_{mat})}{k} = R\left(1 - \frac{E(N_{emp})}{k}\right) = R\left(1 - \prod_{j=0}^{f-1} \frac{D\left(1 - \frac{1}{k}\right) - j}{D - j}\right) \qquad (12)$$

$$Cov(N_{mat}, N_{emp}) \le 0 \qquad \square \qquad (13)$$

In practical scenarios, the data are often sparse, i.e., $f = f_1 + f_2 - a \ll D$. In this case, the upper bound (11) $\left(1 - \frac{1}{k}\right)^f$ is a good approximation to the true value of $\frac{E(N_{emp})}{k}$. Since $\left(1 - \frac{1}{k}\right)^f \approx e^{-f/k}$, we know that the chance of empty bins is small when $f \gg k$. For example, if $f/k = 5$ then $\left(1 - \frac{1}{k}\right)^f \approx 0.0067$. For practical applications, we would expect that $f \gg k$ (for most data pairs), otherwise hashing probably would not be too useful anyway. This is why we do not expect empty bins will significantly impact (if at all) the performance in practical settings.

Lemma 2 shows the following estimator $\hat{R}_{mat}$ of the resemblance is unbiased:

**Lemma 2**

$$\hat{R}_{mat} = \frac{N_{mat}}{k - N_{emp}}, \qquad E\left(\hat{R}_{mat}\right) = R \qquad (14)$$

$$Var\left(\hat{R}_{mat}\right) = R(1-R)\left(E\left(\frac{1}{k - N_{emp}}\right)\left(1 + \frac{1}{f-1}\right) - \frac{1}{f-1}\right) \qquad (15)$$

$$E\left(\frac{1}{k - N_{emp}}\right) = \sum_{j=0}^{k-1} \frac{\mathbf{Pr}(N_{emp} = j)}{k - j} \ge \frac{1}{k - E(N_{emp})} \qquad \square \qquad (16)$$

The fact that $E\left(\hat{R}_{mat}\right) = R$ may seem surprising as in general ratio estimators are not unbiased. Note that $k - N_{emp} > 0$, because we assume the original data vectors are not completely empty (all-zero). As expected, when $k \ll f = f_1 + f_2 - a$, $N_{emp}$ is essentially zero and hence $Var\left(\hat{R}_{mat}\right) \approx \frac{R(1-R)}{k}$. In fact, $Var\left(\hat{R}_{mat}\right)$ is a bit smaller than $\frac{R(1-R)}{k}$, especially for large $k$.

It is probably not surprising that our one permutation scheme (slightly) outperforms the original $k$-permutation scheme (at merely $1/k$ of the preprocessing cost), because one permutation hashing, which is "sampling-without-replacement", provides a better strategy for matrix sparsification.

# 4 Strategies for Dealing with Empty Bins

In general, we expect that empty bins should not occur often because $E(N_{emp})/k \approx e^{-f/k}$, which is very close to zero if $f/k > 5$. (Recall $f = |S_1 \cup S_2|$.) If the goal of using minwise hashing is for data reduction, i.e., reducing the number of nonzeros, then we would expect that $f \gg k$ anyway.

Nevertheless, in applications where we need the estimators to be inner products, we need strategies to deal with empty bins in case they occur. Fortunately, we realize a (in retrospect) simple strategy which can be nicely integrated with linear learning algorithms and performs well.

Figure 3 plots the histogram of the numbers of nonzeros in the *webspam* dataset, which has 350,000 samples. The average number of nonzeros is about 4000 which should be much larger than $k$ (e.g., 500) for the hashing procedure. On the other hand, about 10% (or 2.8%) of the samples have $< 500$ (or $< 200$) nonzeros. Thus, we must deal with empty bins if we do not want to exclude those data points. For example, if $f = k = 500$, then $N_{emp} \approx e^{-f/k} = 0.3679$, which is not small.

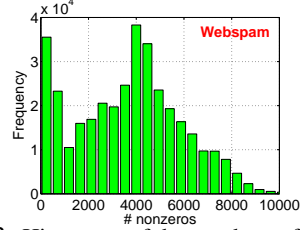

Figure 3: Histogram of the numbers of nonzeros in the *webspam* dataset (350,000 samples).

The strategy we recommend for linear learning is **zero coding**, which is tightly coupled with the strategy of hashed data expansion [16] as reviewed in Sec. 2.2. More details will be elaborated in Sec. 4.2. Basically, we can encode "*" as "zero" in the expanded space, which means $N_{mat}$ will remain the same (after taking the inner product in the expanded space). This strategy, which is **sparsity-preserving**, essentially corresponds to the following modified estimator:

$$\hat{R}_{mat}^{(0)} = \frac{N_{mat}}{\sqrt{k - N_{emp}^{(1)}}\sqrt{k - N_{emp}^{(2)}}} \tag{17}$$

where $N_{emp}^{(1)} = \sum_{j=1}^{k} I_{emp,j}^{(1)}$ and $N_{emp}^{(2)} = \sum_{j=1}^{k} I_{emp,j}^{(2)}$ are the numbers of empty bins in $\pi(S_1)$ and $\pi(S_2)$, respectively. This modified estimator makes sense for a number of reasons.

Basically, since each data vector is processed and coded separately, we actually do not know $N_{emp}$ (the number of *jointly* empty bins) until we see both $\pi(S_1)$ and $\pi(S_2)$. In other words, we can not really compute $N_{emp}$ if we want to use linear estimators. On the other hand, $N_{emp}^{(1)}$ and $N_{emp}^{(2)}$ are always available. In fact, the use of $\sqrt{k - N_{emp}^{(1)}}\sqrt{k - N_{emp}^{(2)}}$ in the denominator corresponds to the normalizing step which is needed before feeding the data to a solver for SVM or logistic regression.

When $N_{emp}^{(1)} = N_{emp}^{(2)} = N_{emp}$, (17) is equivalent to the original $\hat{R}_{mat}$. When two original vectors are very similar (e.g., large $R$), $N_{emp}^{(1)}$ and $N_{emp}^{(2)}$ will be close to $N_{emp}$. When two sets are highly unbalanced, using (17) will overestimate $R$; however, in this case, $N_{mat}$ will be so small that the absolute error will not be large.

## 4.1 The $m$-Permutation Scheme with $1 < m \ll k$

If one would like to further (significantly) reduce the chance of the occurrences of empty bins, here we shall mention that one does not really have to strictly follow "one permutation," since one can always conduct $m$ permutations with $k' = k/m$ and concatenate the hashed data. Once the preprocessing is no longer the bottleneck, it matters less whether we use 1 permutation or (e.g.,) $m = 3$ permutations. The chance of having empty bins decreases exponentially with increasing $m$.

## 4.2 An Example of The "Zero Coding" Strategy for Linear Learning

Sec. 2.2 reviewed the data-expansion strategy used by [16] for integrating $b$-bit minwise hashing with linear learning. We will adopt a similar strategy with modifications for considering empty bins.

We use a similar example as in Sec. 2.2. Suppose we apply our one permutation hashing scheme and use $k = 4$ bins. For the first data vector, the hashed values are $[12013, 25964, 20191, *]$ (i.e., the 4-th bin is empty). Suppose again we use $b = 2$ bits. With the "zero coding" strategy, our procedure

is summarized as follows:

| | | | | |
|---|---|---|---|---|
| Original hashed values ($k = 4$) : | 12013 | 25964 | 20191 | $*$ |
| Original binary representations : | 010111011101101 | 110010101101100 | 100111011011111 | $*$ |
| Lowest $b = 2$ binary digits : | 01 | 00 | 11 | $*$ |
| Expanded $2^b = 4$ binary digits : | 0010 | 0001 | 1000 | 0000 |

New feature vector fed to a solver : $\dfrac{1}{\sqrt{4-1}} \times [0,0,1,0,0,0,0,1,1,0,0,0,0,0,0,0]$

We apply the same procedure to all feature vectors in the data matrix to generate a new data matrix. The normalization factor $\dfrac{1}{\sqrt{k-N_{emp}^{(i)}}}$ varies, depending on the number of empty bins in the $i$-th vector.

## 5   Experimental Results on the Webspam Dataset

The *webspam* dataset has 350,000 samples and 16,609,143 features. Each feature vector has on average about 4000 nonzeros; see Figure 3. Following [16], we use $80\%$ of samples for training and the remaining $20\%$ for testing. We conduct extensive experiments on linear SVM and logistic regression, using our proposed one permutation hashing scheme with $k \in \{2^6, 2^7, 2^8, 2^9\}$ and $b \in \{1, 2, 4, 6, 8\}$. For convenience, we use $D = 2^{24} = 16,777,216$, which is divisible by $k$.

There is one regularization parameter $C$ in linear SVM and logistic regression. Since our purpose is to demonstrate the effectiveness of our proposed hashing scheme, we simply provide the results for a wide range of $C$ values and assume that the best performance is achievable if we conduct cross-validations. This way, interested readers may be able to easily reproduce our experiments.

Figure 4 presents the test accuracies for both linear SVM (upper panels) and logistic regression (bottom panels). Clearly, when $k = 512$ (or even 256) and $b = 8$, $b$-bit one permutation hashing achieves similar test accuracies as using the original data. Also, compared to the original $k$-permutation scheme as in [16], our one permutation scheme achieves similar (or even slightly better) accuracies.

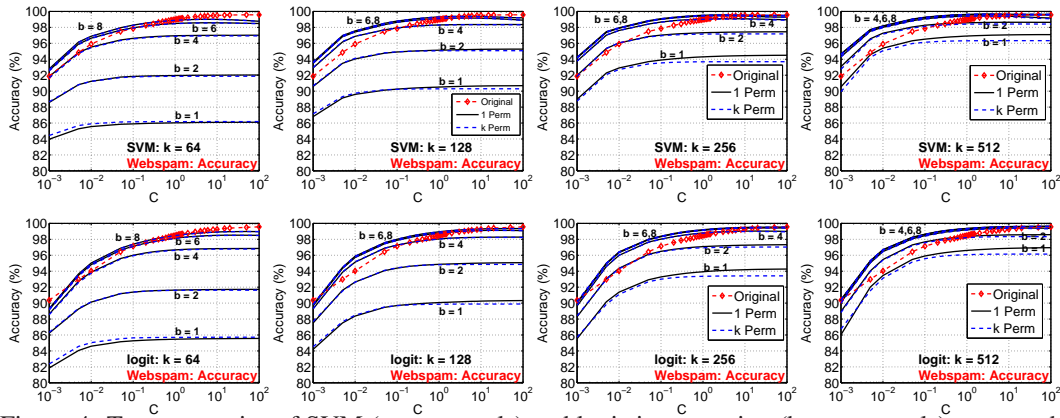

Figure 4: Test accuracies of SVM (upper panels) and logistic regression (bottom panels), averaged over 50 repetitions. The accuracies of using the original data are plotted as dashed (red, if color is available) curves with "diamond" markers. $C$ is the regularization parameter. Compared with the original $k$-permutation minwise hashing (dashed and blue if color is available), the one permutation hashing scheme achieves similar accuracies, or even slightly better accuracies when $k$ is large.

The empirical results on the *webspam* datasets are encouraging because they verify that our proposed one permutation hashing scheme performs as well as (or even slightly better than) the original $k$-permutation scheme, at merely $1/k$ of the original preprocessing cost. On the other hand, it would be more interesting, from the perspective of testing the robustness of our algorithm, to conduct experiments on a dataset (e.g., *news20*) where the empty bins will occur much more frequently.

## 6   Experimental Results on the News20 Dataset

The *news20* dataset (with 20,000 samples and 1,355,191 features) is a very small dataset in not-too-high dimensions. The average number of nonzeros per feature vector is about 500, which is also small. Therefore, this is more like a contrived example and we use it just to verify that our one permutation scheme (with the zero coding strategy) still works very well even when we let $k$ be

as large as 4096 (i.e., most of the bins are empty). In fact, the one permutation schemes achieves noticeably better accuracies than the original $k$-permutation scheme. We believe this is because the one permutation scheme is "sample-without-replacement" and provides a better matrix sparsification strategy without "contaminating" the original data matrix too much.

We experiment with $k \in \{2^5, 2^6, 2^7, 2^8, 2^9, 2^{10}, 2^{11}, 2^{12}\}$ and $b \in \{1, 2, 4, 6, 8\}$, for both one permutation scheme and $k$-permutation scheme. We use 10,000 samples for training and the other 10,000 samples for testing. For convenience, we let $D = 2^{21}$ (which is larger than 1,355,191).

Figure 5 and Figure 6 present the test accuracies for linear SVM and logistic regression, respectively. When $k$ is small (e.g., $k \leq 64$) both the one permutation scheme and the original $k$-permutation scheme perform similarly. For larger $k$ values (especially as $k \geq 256$), however, our one permutation scheme noticeably outperforms the $k$-permutation scheme. Using the original data, the test accuracies are about 98%. Our one permutation scheme with $k \geq 512$ and $b = 8$ essentially achieves the original test accuracies, while the $k$-permutation scheme could only reach about 97.5%.

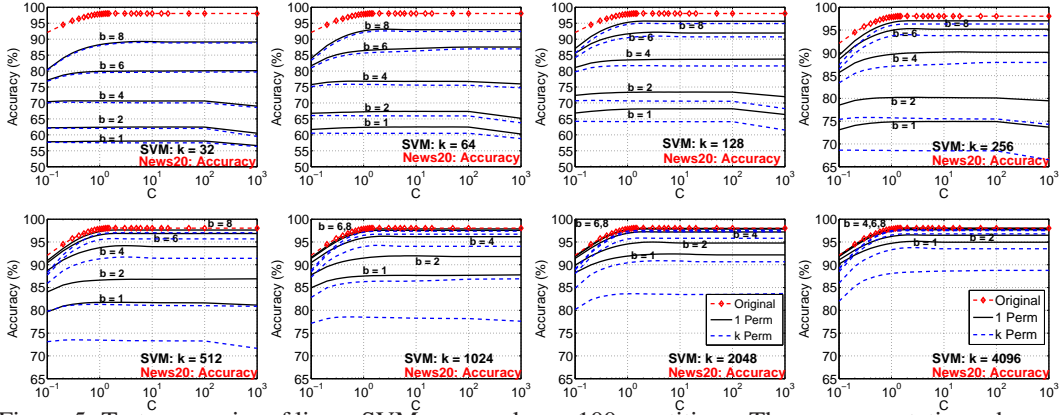

Figure 5: Test accuracies of linear SVM averaged over 100 repetitions. The one permutation scheme noticeably outperforms the original $k$-permutation scheme especially when $k$ is not small.

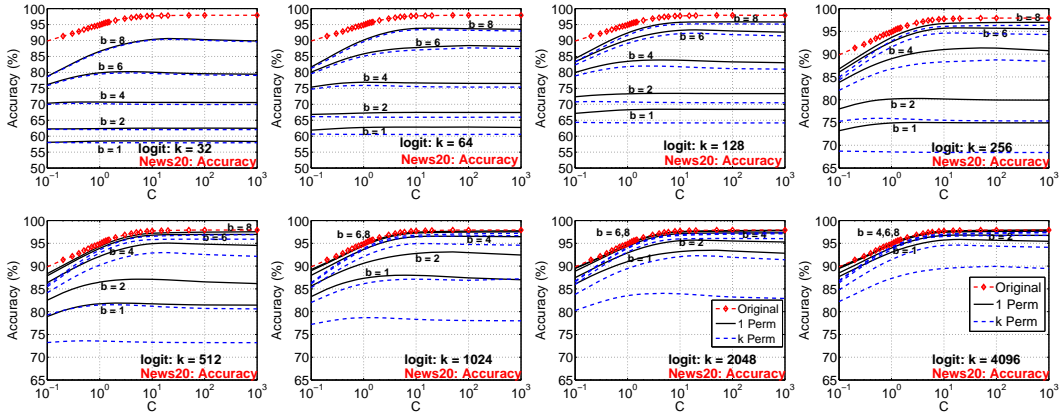

Figure 6: Test accuracies of logistic regression averaged over 100 repetitions. The one permutation scheme noticeably outperforms the original $k$-permutation scheme especially when $k$ is not small.

## 7 Conclusion

A new hashing algorithm is developed for large-scale search and learning in massive binary data. Compared with the original $k$-permutation (e.g., $k = 500$) minwise hashing (which is a standard procedure in the context of search), our method requires only one permutation and can achieve similar or even better accuracies at merely $1/k$ of the original preprocessing cost. We expect that one permutation hashing (or its variant) will be adopted in practice. See more details in arXiv:1208.1259.

**Acknowledgement:** The research of Ping Li is partially supported by NSF-IIS-1249316, NSF-DMS-0808864, NSF-SES-1131848, and ONR-YIP-N000140910911. The research of Art B Owen is partially supported by NSF-0906056. The research of Cun-Hui Zhang is partially supported by NSF-DMS-0906420, NSF-DMS-1106753, NSF-DMS-1209014, and NSA-H98230-11-1-0205.

# References

[1] Alexandr Andoni and Piotr Indyk. Near-optimal hashing algorithms for approximate nearest neighbor in high dimensions. In *Commun. ACM*, volume 51, pages 117–122, 2008.

[2] Leon Bottou. http://leon.bottou.org/projects/sgd.

[3] Andrei Z. Broder, Moses Charikar, Alan M. Frieze, and Michael Mitzenmacher. Min-wise independent permutations (extended abstract). In *STOC*, pages 327–336, Dallas, TX, 1998.

[4] Andrei Z. Broder, Steven C. Glassman, Mark S. Manasse, and Geoffrey Zweig. Syntactic clustering of the web. In *WWW*, pages 1157 – 1166, Santa Clara, CA, 1997.

[5] J. Lawrence Carter and Mark N. Wegman. Universal classes of hash functions (extended abstract). In *STOC*, pages 106–112, 1977.

[6] Graham Cormode and S. Muthukrishnan. An improved data stream summary: the count-min sketch and its applications. *Journal of Algorithm*, 55(1):58–75, 2005.

[7] Rong-En Fan, Kai-Wei Chang, Cho-Jui Hsieh, Xiang-Rui Wang, and Chih-Jen Lin. Liblinear: A library for large linear classification. *Journal of Machine Learning Research*, 9:1871–1874, 2008.

[8] Dennis Fetterly, Mark Manasse, Marc Najork, and Janet L. Wiener. A large-scale study of the evolution of web pages. In *WWW*, pages 669–678, Budapest, Hungary, 2003.

[9] Jerome H. Friedman, F. Baskett, and L. Shustek. An algorithm for finding nearest neighbors. *IEEE Transactions on Computers*, 24:1000–1006, 1975.

[10] Piotr Indyk and Rajeev Motwani. Approximate nearest neighbors: Towards removing the curse of dimensionality. In *STOC*, pages 604–613, Dallas, TX, 1998.

[11] Thorsten Joachims. Training linear svms in linear time. In *KDD*, pages 217–226, Pittsburgh, PA, 2006.

[12] Ping Li. Very sparse stable random projections for dimension reduction in $l_\alpha$ ($0 < \alpha \leq 2$) norm. In *KDD*, San Jose, CA, 2007.

[13] Ping Li and Kenneth W. Church. Using sketches to estimate associations. In *HLT/EMNLP*, pages 708–715, Vancouver, BC, Canada, 2005 (The full paper appeared in Commputational Linguistics in 2007).

[14] Ping Li, Kenneth W. Church, and Trevor J. Hastie. One sketch for all: Theory and applications of conditional random sampling. In *NIPS*, Vancouver, BC, Canada, 2008 (Preliminary results appeared in NIPS 2006).

[15] Ping Li, Trevor J. Hastie, and Kenneth W. Church. Very sparse random projections. In *KDD*, pages 287–296, Philadelphia, PA, 2006.

[16] Ping Li, Anshumali Shrivastava, Joshua Moore, and Arnd Christian König. Hashing algorithms for large-scale learning. In *NIPS*, Granada, Spain, 2011.

[17] Ping Li, Anshumali Shrivastava, and Arnd Christian König. b-bit minwise hashing in practice: Large-scale batch and online learning and using GPUs for fast preprocessing with simple hash functions. Technical report.

[18] Ping Li and Arnd Christian König. b-bit minwise hashing. In *WWW*, pages 671–680, Raleigh, NC, 2010.

[19] Ping Li, Arnd Christian König, and Wenhao Gui. b-bit minwise hashing for estimating three-way similarities. In *NIPS*, Vancouver, BC, 2010.

[20] Shai Shalev-Shwartz, Yoram Singer, and Nathan Srebro. Pegasos: Primal estimated sub-gradient solver for svm. In *ICML*, pages 807–814, Corvalis, Oregon, 2007.

[21] Qinfeng Shi, James Petterson, Gideon Dror, John Langford, Alex Smola, and S.V.N. Vishwanathan. Hash kernels for structured data. *Journal of Machine Learning Research*, 10:2615–2637, 2009.

[22] Anshumali Shrivastava and Ping Li. Fast near neighbor search in high-dimensional binary data. In *ECML*, 2012.

[23] Josef Sivic and Andrew Zisserman. Video google: a text retrieval approach to object matching in videos. In *ICCV*, 2003.

[24] Simon Tong. Lessons learned developing a practical large scale machine learning system. http://googleresearch.blogspot.com/2010/04/lessons-learned-developing-practical.html, 2008.

[25] Kilian Weinberger, Anirban Dasgupta, John Langford, Alex Smola, and Josh Attenberg. Feature hashing for large scale multitask learning. In *ICML*, pages 1113–1120, 2009.

